# A Neural Network Classifier Based on Coding Theory

Tzi-Dar Chiueh and Rodney Goodman
California Institute of Technology, Pasadena, California 91125

## ABSTRACT

The new neural network classifier we propose transforms the classification problem into the coding theory problem of decoding a noisy codeword. An input vector in the feature space is transformed into an internal representation which is a codeword in the code space, and then error correction decoded in this space to classify the input feature vector to its class. Two classes of codes which give high performance are the Hadamard matrix code and the maximal length sequence code. We show that the number of classes stored in an N-neuron system is linear in N and significantly more than that obtainable by using the Hopfield type memory as a classifier.

## I. INTRODUCTION

Associative recall using neural networks has recently received a great deal of attention. Hopfield in his papers [1,2] describes a mechanism which iterates through a feedback loop and stabilizes at the memory element that is nearest the input, provided that not many memory vectors are stored in the machine. He has also shown that the number of memories that can be stored in an N-neuron system is about $0.15N$ for N between 30 and 100. McEliece et al. in their work [3] showed that for synchronous operation of the Hopfield memory about $N/(2\log N)$ data vectors can be stored reliably when N is large. Abu-Mostafa [4] has predicted that the upper bound for the number of data vectors in an N-neuron Hopfield machine is N. We believe that one should be able to devise a machine with M, the number of data vectors, *linear* in N and larger than the $0.15N$ achieved by the Hopfield method.

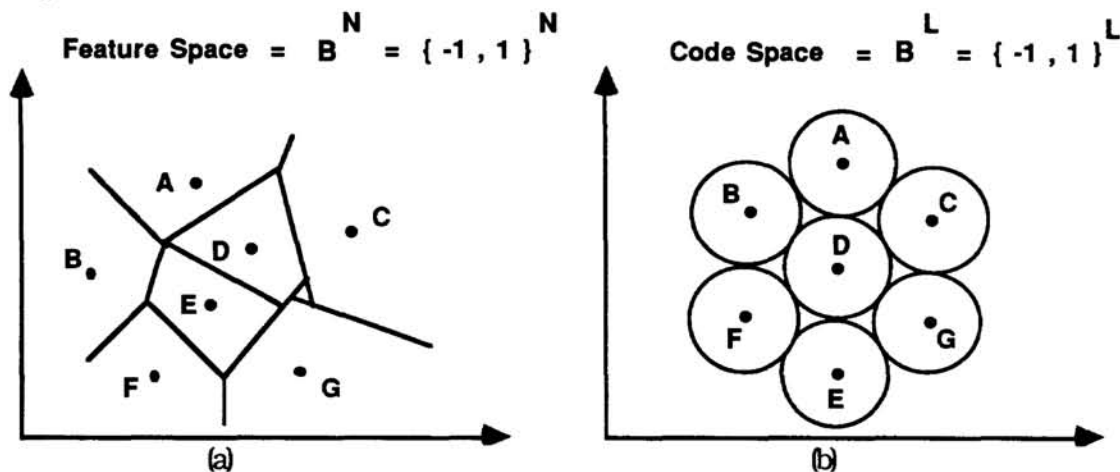

**Figure 1** (a) Classification problems versus (b) Error control decoding problems

In this paper we are specifically concerned with the problem of classification as in pattern recognition. We propose a new method of building a neural network classifier, based on the well established techniques of error control coding. Consider a typical classification problem (Fig. 1(a)), in which one is given *a priori* a set of classes, $C^{(\alpha)}$, $\alpha = 1, \ldots, M$. Associated with each class is a feature vector which labels the class ( the *exemplar* of the class ), i.e. it is the

most representative point in the class region. The input is classified into the class with the nearest exemplar to the input. Hence for each class there is a region in the N-dimensional binary feature space $\mathbf{B}^N \equiv \{1,-1\}^N$, in which every vector will be classified to the corresponding class.

A similar problem is that of decoding a codeword in an error correcting code as shown in Fig. 1(b). In this case codewords are constructed by design and are usually at least $d_{min}$ apart. The received corrupted codeword is the input to the decoder, which then finds the nearest codeword to the input. In principle then, if the distance between codewords is greater than $2t+1$, it is possible to decode (or classify) a noisy codeword (feature vector) into the correct codeword (exemplar) provided that the Hamming distance between the noisy codeword and the correct codeword is no more than $t$. Note that there is no guarantee that the exemplars are uniformly distributed in $\mathbf{B}^N$, consequently the *attraction radius* (the maximum number of errors that can occur in any given feature vector such that the vector can still be correctly classified) will depend on the *minimum* distance between exemplars.

Many solutions to the minimum Hamming distance classification have been proposed, the one commonly used is derived from the idea of matched filters in communication theory. Lippmann [5] proposed a two-stage neural network that solves this classification problem by first correlating the input with all exemplars and then picking the maximum by a "winner-take-all" circuit or a network composed of two-input comparators. In Figure 2, $f_1, f_2, ..., f_N$ are the N input bits, and $s_1, s_2, ..., s_M$ are the matching scores(similarity) of $\mathbf{f}$ with the M exemplars. The second block picks the maximum of $s_1, s_2, ..., s_M$ and produces the index of the exemplar with the largest score. The main disadvantage of such a classifier is the complexity of the maximum-picking circuit, for example a "winner-take-all" net needs connection weights of large dynamic range and graded-response neurons, whilst the comparator maximum net demands M-1 comparators organized in $\log_2 M$ stages.

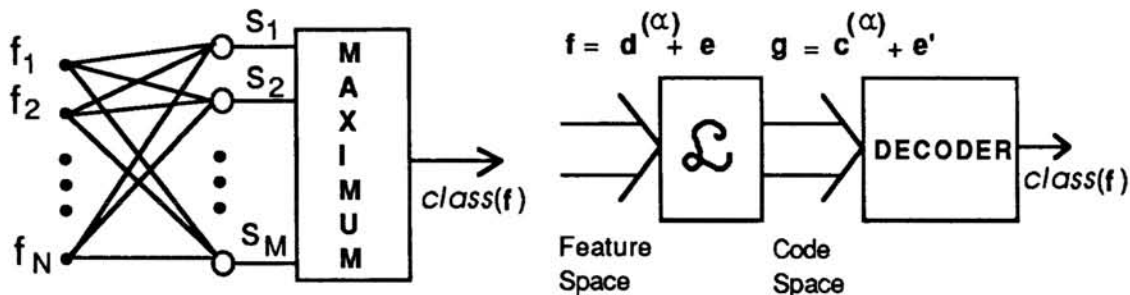

Fig. 2 A matched filter type classifier    Fig. 3 Structure of the proposed classifier

Our main idea is thus to transform every vector in the feature space to a vector in some code space in such a way that every exemplar corresponds to a codeword in that code. The code should preferably (but not necessarily) have the property that codewords are uniformly distributed in the code space, that is, the Hamming distance between every pair of codewords is the same. With this transformation, we turn the problem of classification into the coding problem of decoding a noisy codeword. We then do error correction decoding on the vector in the code space to obtain the index of the noisy codeword and hence classify the original feature vector, as shown in Figure 3.

This paper develops the construction of such a classification machine as follows. First we consider the problem of transforming the input vectors from the feature space to the code space. We describe two hetero-associative memories for doing this, the first method uses an outer product matrix technique similar to

that of Hopfield's, and the second method generates its matrix by the pseudoinverse technique[6,7]. Given that we have transformed the problem of associative recall, or classification, into the problem of decoding a noisy codeword, we next consider suitable codes for our machine. We require the codewords in this code to have the property of orthogonality or *pseudo*-orthogonality, that is, the ratio of the cross-correlation to the auto-correlation of the codewords is small. We show two classes of such good codes for this particular decoding problem i.e. the Hadamard matrix codes, and the maximal length sequence codes[8]. We next formulate the complete decoding algorithm, and describe the overall structure of the classifier in terms of a two layer neural network. The first layer performs the mapping operation on the input, and the second one decodes its output to produce the index of the class to which the input belongs.

The second part of the paper is concerned with the performance of the classifier. We first analyze the performance of this new classifier by finding the relation between the maximum number of classes that can be stored and the classification error rate. We show (when using a transform based on the outer product method) that for negligible misclassification rate and large N, a not very tight lower bound on M, the number of stored classes, is 0.22N. We then present comprehensive simulation results that confirm and exceed our theoretical expectations. The simulation results compare our method with the Hopfield model for both the outer product and pseudo-inverse method, and for both the analog and hard limited connection matrices. In all cases our classifier exceeds the performance of the Hopfield memory in terms of the number of classes that can be reliably recovered.

## II. TRANSFORM TECHNIQUES

Our objective is to build a machine that can discriminate among input vectors and classify each one of them into the appropriate class. Suppose $d^{(\alpha)} \in B^N$ is the exemplar of the corresponding class $C^{(\alpha)}$, $\alpha = 1, 2, \ldots, M$. Given the input $f$, we want the machine to be able to identify the class whose exemplar is closest to $f$, that is, we want to calculate the following function,

$$class(f) = \alpha \qquad iff \quad |f - d^{(\alpha)}| < |f - d^{(\beta)}| \qquad \forall \alpha \neq \beta$$

where $| \ |$ denotes Hamming distance in $B^N$.

We approach the problem by seeking a transform $\mathcal{L}$ that maps each exemplar $d^{(\alpha)}$ in $B^N$ to the corresponding codeword $w^{(\alpha)}$ in $B^L$. And an input feature vector $f = d^{(\gamma)} + e$ is thus mapped to a noisy codeword $g = w^{(\gamma)} + e'$ where $e$ is the error added to the exemplar, and $e'$ is the corresponding error pattern in the code space. We then do error correction decoding on $g$ to get the index of the corresponding codeword. Note that $e'$ may not have the same Hamming weight as $e$, that is, the transformation $\mathcal{L}$ may either generate more errors or eliminate errors that are present in the original input feature vector. We require $\mathcal{L}$ to satisfy the following equation,

$$\mathcal{L}d^{(\alpha)} = w^{(\alpha)} \qquad \alpha = 0, 1, \ldots, M-1$$

and $\mathcal{L}$ will be implemented using a single-layer feedforward network.

Thus we first construct a matrix according to the sets of $d^{(\alpha)}$'s and $w^{(\alpha)}$'s, call it $T$, and define $\mathcal{L}$ as

$$\mathcal{L} \equiv sgn \circ T$$

where $sgn$ is the threshold operator that maps a vector in $R^L$ to $B^L$ and $R$ is the field of real numbers.

Let $D$ be an N x M matrix whose $\alpha$th column is $d^{(\alpha)}$ and $W$ be an L x M matrix whose $\beta$th column is $w^{(\beta)}$. The two possible methods of constructing the matrix for $\mathcal{L}$ are as follows:

*Scheme A (outer product method)* [3,6] : In this scheme the matrix $T$ is defined as the sum of outer products of all exemplar-codeword pairs, i.e.

$$T^{(A)}{}_{ij} = \sum_{\alpha=0}^{M-1} w_i(\alpha) \cdot d_j(\alpha)$$

or equivalently,

$$T^{(A)} = W D^t$$

*Scheme B (pseudo-inverse method)* [6,7] : We want to find a matrix $T^{(B)}$ satisfying the following equation,

$$T^{(B)} D = W$$

In general $D$ is not a square matrix, moreover $D$ may be singular, so $D^{-1}$ may not exist. To circumvent this difficulty, we calculate the *pseudo-inverse* (denoted $D^\dagger$) of the matrix $D$ instead of its real inverse, let $D^\dagger \equiv (D^tD)^{-1}D^t$. $T^{(B)}$ can be formulated as,

$$T^{(B)} = W D^\dagger = W (D^t D)^{-1} D^t$$

### III. CODES

The codes we are looking for should preferably have the property that its codewords be distributed uniformly in $B^L$, that is, the distance between each two codewords must be the same and as large as possible. We thus seek classes of *equidistant* codes. Two such classes are the Hadamard matrix codes, and the maximal length sequence codes.

First define the word *pseudo-orthogonal* .

*Definition* : Let $w^{(\alpha)} = (w_0^{(\alpha)}, w_1^{(\alpha)}, \ldots , w_{L-1}^{(\alpha)}) \in B^L$ be the $\alpha$th codeword of code C, where $\alpha = 1, 2, \ldots , M$. Code C is said to be *pseudo-orthogonal* iff

$$(w^{(\alpha)}, w^{(\beta)}) = \sum_{i=0}^{L-1} w_i^{(\alpha)} w_i^{(\beta)}$$
$$= \begin{cases} L & \alpha = \beta \\ \epsilon & \alpha \neq \beta \end{cases} \qquad \text{where } \epsilon << L$$

where ( , ) denotes inner product of two vectors.

**Hadamard Matrices:** An orthogonal code of length L whose L codewords are rows or columns of an L x L Hadamard matrix. In this case $\epsilon = 0$ and the distance between any two codewords is L/2. It is conjectured that there exist such codes for all L which are multiples of 4, thus providing a large class of codes[8].

**Maximal Length Sequence Codes:** There exists a family of maximal length sequence (also called pseudo-random or PN sequence) codes[8], generated by shift registers, that satisfy pseudo-orthogonality with $\epsilon = -1$. Suppose $g(x)$ is a primitive polynomial over $GF(2)$ of degree D, and let $L = 2^D - 1$, and if

$$f(x) = 1/g(x) = \sum_{k=0}^{\infty} c_k \cdot x^k$$

then $c_0, c_1, \ldots\ldots$ is a periodic sequence of period L ( since $g(x) \mid x^L - 1$). If code C is made up of the L cyclic shifts of

$$\mathbf{c} = (1 - 2c_0, 1 - 2c_1, \ldots, 1 - 2c_{L-1})$$

then code C satisfies pseudo-orthogonality with $\epsilon = -1$. One then easily sees that the minimum distance of this code is $(L - 1)/2$ which gives a correcting power of approximately $L/4$ errors for large L.

## IV. OVERALL CLASSIFIER STRUCTURE

We shall now describe the overall classifier structure, essentially it consists of the mapping $\mathcal{L}$ followed by the error correction decoder for the maximal length sequence code or Hadamard matrix code. The decoder operates by correlating the input vector with every codeword and then thresholding the result at $(L + \epsilon)/2$. The rationale of this algorithm is as follows, since the distance between every two codewords in this code is exactly $(L - \epsilon)/2$ bits, the decoder should be able to correct any error pattern with less than $(L - \epsilon)/4$ errors if the threshold is set halfway between L and $\epsilon$ i.e. $(L + \epsilon)/2$.

Suppose the input vector to the decoder is $\mathbf{g} = \mathbf{w}^{(\alpha)} + \mathbf{e}$ and $\mathbf{e}$ has Hamming weight s (i.e. s nonzero components) then we have

$$(\mathbf{g}, \mathbf{w}^{(\alpha)}) = L - 2s$$
$$(\mathbf{g}, \mathbf{w}^{(\beta)}) \leq 2s + \epsilon \qquad \text{where } \beta \neq \alpha$$

From the above equation, if $\mathbf{g}$ is less than $(L - \epsilon)/4$ errors away from $\mathbf{w}^{(\alpha)}$ (i.e. $s < (L - \epsilon)/4$) then $(\mathbf{g}, \mathbf{w}^{(\alpha)})$ will be more than $(L + \epsilon)/2$ and $(\mathbf{g}, \mathbf{w}^{(\beta)})$ will be less than $(L + \epsilon)/2$, for all $\beta \neq \alpha$. As a result, we arrive at the following decoding algorithm,

$$decode(\mathbf{g}) = sgn(\mathbf{W}^t \mathbf{g} - ((L + \epsilon)/2)\mathbf{j})$$

where $\mathbf{j} = [1\ 1\ \ldots\ 1]^t$, which is an M x 1 vector.

In the case when $\epsilon = -1$ and less than $(L+1)/4$ errors in the input, the output will be a vector in $\mathbf{B}^M \equiv \{1, -1\}^M$ with only *one* component positive (+1), the index of which is the index of the class that the input vector belongs. However if there are more than $(L+1)/4$ errors, the output can be either the all negative(-1) vector (decoder failure) or another vector with one positive component(decoder error).

The function *class* can now be defined as the composition of $\mathcal{L}$ and *decode*, the overall structure of the new classifier is depicted in Figure 4. It can be viewed as a two-layer neural network with L hidden units and M output neurons. The first layer is for mapping the input feature vector to a noisy codeword in the code space ( the "internal representation" ) while the second one decodes the first's output and produces the index of the class to which the input belongs.

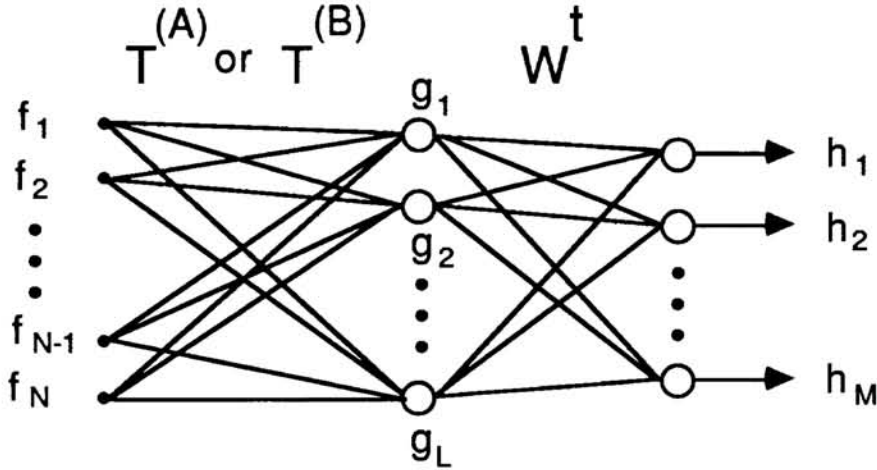

Figure 4    Overall architecture of the new neural network classifier

## V.  PERFORMANCE ANALYSIS

From the previous section, we know that our classifier will make an error only if the transformed vector in the code space, which is the input to the decoder, has no less than $(L - \epsilon)/4$ errors.  We now proceed to find the error rate for this classifier in the case when the input is one of the exemplars (i.e. no error), say $\mathbf{f} = \mathbf{d}^{(\beta)}$ and an outer product connection matrix for $\mathcal{L}$.  Following the approach of McEliece et. al.[3], we have

$$(\mathcal{L}\,\mathbf{d}^{(\beta)})_i = sgn\,(\sum_{j=0}^{N-1}\sum_{\alpha=0}^{M-1} w_i^{(\alpha)}\,d_j^{(\alpha)}\,d_j^{(\beta)}\,)$$

$$= sgn\,(\,N\,w_i^{(\beta)} + \sum_{\substack{j=0 \\ }}^{N-1}\sum_{\substack{\alpha=0 \\ \alpha\neq\beta}}^{M-1} w_i^{(\alpha)}\,d_j^{(\alpha)}\,d_j^{(\beta)}\,)$$

Assume without loss of generality that $w_i^{(\beta)} = -1$, and if

$$X \equiv \sum_{\substack{j=0 \\ }}^{N-1}\sum_{\substack{\alpha=0 \\ \alpha\neq\beta}}^{M-1} w_i^{(\alpha)}\,d_j^{(\alpha)}\,d_j^{(\beta)} \geq N$$

then

$$(\mathcal{L}\,\mathbf{d}^{(\beta)})_i \neq w_i^{(\beta)}$$

Notice that we assumed all $\mathbf{d}^{(\alpha)}$'s are random, namely each component of any $\mathbf{d}^{(\alpha)}$ is the outcome of a Bernoulli trial, accordingly, X is the sum of $N(M-1)$ independent identically distributed random variables with mean 0 and variance 1.   In the asymptotic case, when N and M are both very large, X can be approximated by a normal distribution with mean 0, variance NM.  Thus

$$p \equiv Pr\{\,(\mathcal{L}\,\mathbf{d}^{(\beta)})_i \neq w_i^{(\beta)}\,\}$$

$$\simeq Q(\sqrt{N/M})$$

$$\text{where } Q(x) = \frac{1}{\sqrt{2\pi}}\int_x^\infty e^{t^2/2}\,dt$$

Next we calculate the misclassification rate of the new classifier as follows (assuming $\epsilon \ll L$),

$$P_e = \sum_{k=\lfloor L/4 \rfloor}^{L} \binom{L}{k} p^k (1-p)^{L-k}$$

where $\lfloor \ \rfloor$ is the integer floor. Since in general it is not possible to express the summation explicitly, we use the Chernoff method to bound $P_e$ from above. Multiplying each term in the summation by a number larger than unity ( $e^{t(k-L/4)}$ with $t > 0$ ) and summing from $k = 0$ instead of $k = \lfloor L/4 \rfloor$.

$$P_e < \sum_{k=0}^{L} \binom{L}{k} p^k (1-p)^{L-k} e^{t(k-L/4)} = e^{-Lt/4}(1-p+pe^t)^L$$

Differentiating the RHS of the above equation w.r.t. t and set it to 0, we find the optimal $t_0$ as $e^{t_0} = (1-p)/3p$. The condition that $t_0 > 0$ implies that $p < 1/4$, and since we are dealing with the case where p is small, it is automatically satisfied. Substituting the optimal $t_0$, we obtain

$$P_e < c^L \cdot p^{L/4} \cdot (1-p)^{3L/4} \qquad \text{where } c = 4/(3^{3/4}) = 1.7547654$$

From the expression for $P_e$ , we can estimate M, the number of classes that can be classified with negligible misclassification rate, in the following way, suppose $P_e = \delta$ where $\delta \ll 1$ and $p \ll 1$, then

$$\delta^{4/L} < c^4 \cdot p \cdot (1-p)^3 \quad \Rightarrow \quad p = Q(\sqrt{N/M}) > c^{-4} \cdot (1-p)^{-3} \cdot \delta^{4/L}$$

For small x we have $Q^{-1}(x) \sim \sqrt{2 log(1/x)}$ and since $\delta$ is a fixed value, as L approaches infinity, we have

$$M > \frac{N}{8 \log c} = \frac{N}{4.5}$$

From the above lower bound for M, one easily see that this new machine is able to classify a constant times N classes, which is better than the number of memory items a Hopfield model can store i.e. N/(2logN). Although the analysis is done assuming N approaches infinity, the simulation results in the next section show that when N is moderately large (e.g. 63) the above lower bound applies.

## VI. SIMULATION RESULTS AND A CHARACTER RECOGNITION EXAMPLE

We have simulated both the Hopfield model and our new machine(using maximal length sequence codes) for L = N = 31, 63 and for the following four cases respectively.
(i) connection matrix generated by outer product method
(ii) connection matrix generated by pseudo-inverse method
(iii) connection matrix generated by outer product method, the components of the connection matrix are hard limited.
(iv) connection matrix generated by pseudo-inverse method, the components of the connection matrix are hard limited.

For each case and each choice of N, the program fixes M and the number of errors in the input vector, then randomly generates 50 sets of M exemplars and computes the connection matrix for each machine. For each machine it randomly picks an exemplar and adds noise to it by randomly complementing the specified number of bits to generate 20 trial input vectors, it then simulates the machine and checks whether or not the input is classified to the nearest class and reports the percentage of success for each machine.

The simulation results are shown in Figure 5, in each graph the horizontal axis is M and the vertical axis is the attraction radius. The data we show are obtained by collecting only those cases when the success rate is more than 98%, that is for fixed M what is the largest attraction radius (number of bits in error of the input vector) that has a success rate of more than 98%. Here we use the attraction radius of -1 to denote that for this particular M, with the input being an exemplar, the success rate is less than 98% in that machine.

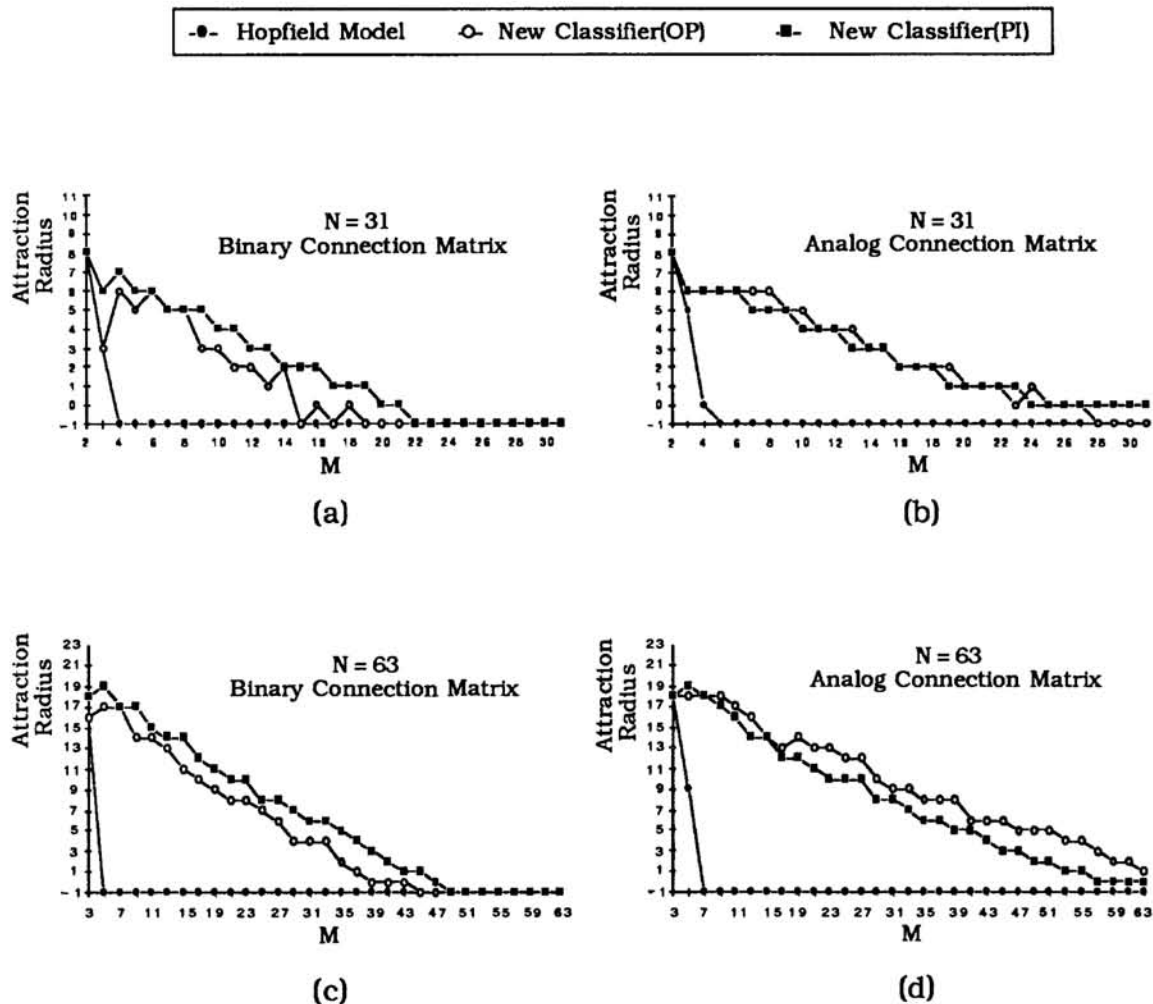

Figure 5 Simulation results of the Hopfield memory and the new classifier

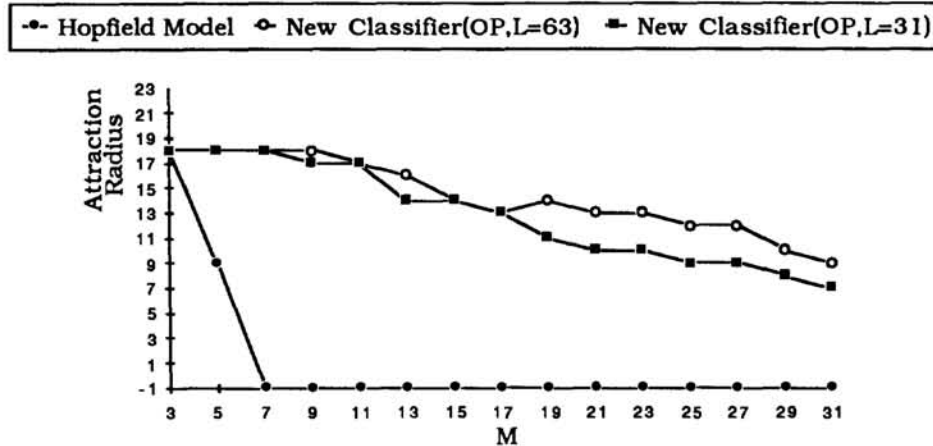

Figure 6  Performance of the new classifier using codes of different lengths

In all cases our classifier exceeds the performance of the Hopfield model in terms of the number of classes that can be reliably recovered. For example, consider the case of N = 63 and a hard limited connection matrix for both the new classifier and the Hopfield model, we find that for an attraction radius of zero, that is, no error in the input vector, the Hopfield model has a classification capacity of approximately 5, while our new model can store 47. Also, for an attraction radius of 8, that is, an average of N/8 errors in the input vector, the Hopfield model can reliably  store 4 classes while our new model stores 27 classes. Another simulation (Fig. 6) using a shorter code (L = 31 instead of L = 63) reveals that by shortening the code, the performance of the classifier degrades only slightly.  We therefore conjecture that it is possible to use traditional error correcting codes (e.g. BCH code) as internal representations, however, by going to a higher rate code, one is trading minimum distance of the code (error tolerance) for complexity (number of hidden units), which implies possibly poorer performance of the classifier.

We also notice that the superiority of the pseudoinverse method over the outer product method appears only when the connection matrices are hard limited.   The reason for this is that the pseudoinverse method is best for decorrelating the dependency among exemplars, yet the exemplars in this simulation are generated randomly and are presumably independent, consequently one can not see the advantage of pseudoinverse method.   For correlated  exemplars, we expect the pseudoinverse method to be clearly better (see next example).

Next we present an example of applying this classifier to recognizing characters.  Each character is represented by a 9 x 7 pixel array, the input is generated by flipping every pixel with 0.1 and 0.2 probability.  The input is then passed to five machines:  Hopfield memory, the new classifier with either the pseudoinverse method or outer product method, and L = 7 or L = 31.  Figure 7 and 8 show the results of all 5 machines for 0.1 and 0.2 pixel flipping probability respectively, a blank output means that the classifier refuses to make a decision. First note that the L = 7 case is not necessarily worse than the  L = 31 case, this confirms the earlier conjecture that fewer hidden units (shorter code) only degrades performance slightly.  Also one easily sees that the pseudoinverse method is better than the outer product method because of the correlation between exemplars.  Both methods outperform the Hopfield memory since the latter mixes exemplars that are to be remembered and produces a blend of exemplars rather than the exemplars themselves, accordingly it cannot classify the input without mistakes.

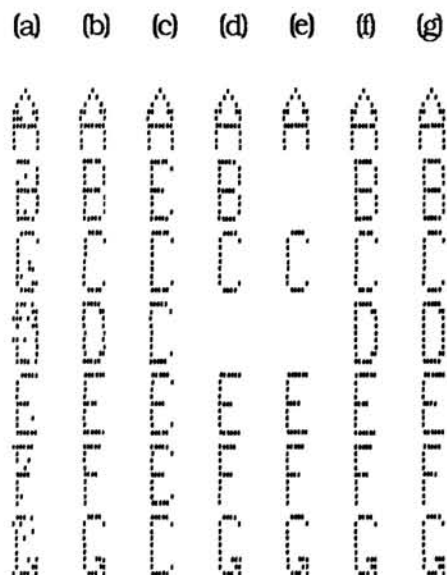
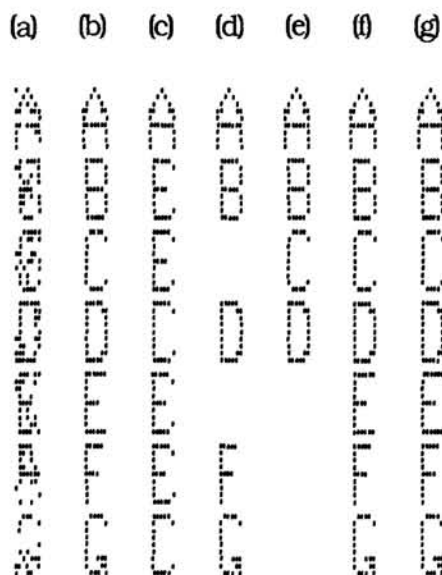

Figure 7 The character recognition example with 10% pixel reverse probability (a) input (b) correct output (c) Hopfield Model (d)-(g) new classifier (d) OP, L = 7 (e)OP, L = 31 (f) PI, L = 7 (g) PI, L = 31

Figure 8 The character recognition example with 20% pixel reverse probability (a) input (b) correct output (c) Hopfield Model (d)-(g) new classifier (d) OP, L = 7 (e)OP, L = 31 (f) PI, L = 7 (g) PI, L = 31

## VII. CONCLUSION

In this paper we have presented a new neural network classifier design based on coding theory techniques. The classifier uses codewords from an error correcting code as its internal representations. Two classes of codes which give high performance are the Hadamard matrix codes and the maximal length sequence codes. In performance terms we have shown that the new machine is significantly better than using the Hopfield model as a classifier. We should also note that when comparing the new classifier with the Hopfield model, the increased performance of the new classifier does not entail extra complexity, since it needs only $L + M$ hard limiter neurons and $L(N + M)$ connection weights versus N neurons and $N^2$ weights in a Hopfield memory.

In conclusion we believe that our model forms the basis of a fast, practical method of classification with an efficiency greater than other previous neural network techniques.

## REFERENCES

[1] J. J. Hopfield, *Proc. Nat. Acad. Sci. USA* , Vol. 79, pp. 2554-2558 (1982).

[2] J. J. Hopfield, *Proc. Nat. Acad. Sci. USA* , Vol. 81, pp. 3088-3092 (1984).

[3] R. J. McEliece, et. al, *IEEE Tran. on Information Theory* , Vol. IT-33, pp. 461-482 (1987).

[4] Y. S. Abu-Mostafa and J. St. Jacques, *IEEE Tran. on Information Theory* , Vol. IT-31, pp. 461-464 (1985).

[5] R. Lippmann, *IEEE ASSP Magazine* , Vol. 4, No. 2, pp. 4-22 (April 1987).

[6] T. Kohonen, *Associative Memory - A System-Theoretical Approach* (Springer-Verlag, Berlin Heidelberg, 1977).

[7] S. S. Venkatesh,*Linear Map with Point Rules* , Ph. D Thesis, Caltech, 1987.

[8] E. R. Berlekamp, *Algebraic Coding Theory* , Aegean Park Press, 1984.
